# Bayesian Optimization with a Finite Budget: An Approximate Dynamic Programming Approach

**Remi R. Lam**
Massachusetts Institute of Technology
Cambridge, MA
rlam@mit.edu

**Karen E. Willcox**
Massachusetts Institute of Technology
Cambridge, MA
kwillcox@mit.edu

**David H. Wolpert**
Santa Fe Institute
Santa Fe, NM
dhw@santafe.edu

## Abstract

We consider the problem of optimizing an expensive objective function when a finite budget of total evaluations is prescribed. In that context, the optimal solution strategy for Bayesian optimization can be formulated as a dynamic programming instance. This results in a complex problem with uncountable, dimension-increasing state space and an uncountable control space. We show how to approximate the solution of this dynamic programming problem using rollout, and propose rollout heuristics specifically designed for the Bayesian optimization setting. We present numerical experiments showing that the resulting algorithm for optimization with a finite budget outperforms several popular Bayesian optimization algorithms.

## 1 Introduction

Optimizing an objective function is a central component of many algorithms in machine learning and engineering. It is also essential to many scientific models, concerning everything from human behavior, to protein folding, to population biology. Often, the objective function to optimize is non-convex and does not have a known closed-form expression. In addition, the evaluation of this function can be expensive, involving a time-consuming computation (e.g., training a neural network, numerically solving a set of partial differential equations, etc.) or a costly experiment (e.g., drilling a borehole, administering a treatment, etc.). Accordingly, there is often a *finite budget* specifying the maximum number of evaluations of the objective function allowed to perform the optimization.

Bayesian optimization (BO) has become a popular optimization technique for solving problems governed by such expensive objective functions [17, 9, 2]. BO iteratively updates a statistical model and uses it as a surrogate for the objective function. At each iteration, this statistical model is used to select the next design to evaluate. Most BO algorithms are *greedy*, ignoring how the design selected at a given iteration will affect the future steps of the optimization. Thus, the decisions made are typically one-step optimal. Because of this shortsightedness, such algorithms balance, in a greedy fashion, the BO exploration-exploitation trade-off: evaluating designs to improve the statistical model or to find the optimizer of the objective function.

In contrast to greedy algorithms, a *lookahead* approach is aware of the remaining evaluations and can balance the exploration-exploitation trade-off in a principled way. A lookahead approach builds an optimal strategy that maximizes a *long-term reward* over several steps. That optimal strategy is the solution of a challenging dynamic programming (DP) problem whose complexity stems, in part, from

the increasing dimensionality of the involved spaces as the budget increases, and from the presence of nested maximizations and expectations. This is especially challenging when the design space takes an uncountable set of values.

The first contribution of this paper is to use rollout [1], an approximate dynamic programming (ADP) algorithm to circumvent the nested maximizations of the DP formulation. This leads to a problem significantly simpler to solve. Rollout uses suboptimal heuristics to guide the simulation of optimization scenarios over several steps. Those simulations allow us to quantify the long-term benefits of evaluating a given design. The heuristics used by rollout are typically problem-dependent. The second contribution of this paper is to build heuristics adapted to BO with a finite budget that leverage existing greedy BO strategies. As demonstrated with numerical experiments, this can lead to improvements in performance.

The following section of this paper provides a brief description of Gaussian processes and their use in Bayesian optimization (Sec. 2), followed by a brief overview of dynamic programming (Sec. 3). Sec. 4 develops the connection between BO and DP and discusses some of the related work. We then propose to employ the rollout algorithm (with heuristics adapted to BO) to mitigate the complexity of the DP algorithm (Sec. 5). In Sec. 6, we numerically investigate the proposed algorithm and present our conclusions in Sec. 7.

## 2  Bayesian Optimization

We consider the following optimization problem:

$$\text{(OP)} \quad \boldsymbol{x}^* = \text{argmin}_{\boldsymbol{x} \in \mathcal{X}} \, f(\boldsymbol{x}), \tag{1}$$

where $\boldsymbol{x}$ is a $d$-dimensional vector of design variables. The design space, $\mathcal{X}$, is a bounded subset of $\mathbb{R}^d$, and $f : \mathcal{X} \mapsto \mathbb{R}$ is an objective function that is expensive to evaluate. We are interested in finding a minimizer $\boldsymbol{x}^*$ of the objective function using a *finite budget* of $N$ function evaluations. We refer to this problem as the *original problem* (OP).

In the Bayesian optimization (BO) setting, the (deterministic or noisy) objective function $f$ is modeled as a realization of a stochastic process, typically a Gaussian process (GP) $\mathcal{G}$, on a probability space $(\Omega, \Sigma, \mathbb{P})$, which defines a prior distribution over functions. A GP is fully defined by a mean function $m : \mathcal{X} \to \mathbb{R}$ (often set to zero without loss of generality) and a covariance kernel $\kappa : \mathcal{X}^2 \to \mathbb{R}$ (see [16] for an overview of GP):

$$f \sim \mathcal{G}(m, \kappa). \tag{2}$$

The BO algorithm starts with an initial design $\boldsymbol{x}_1$ and its associated value $y_1 = f(\boldsymbol{x}_1)$ provided by the user. This defines the first training set $\mathcal{S}_1 = \{(\boldsymbol{x}_1, y_1)\}$. At each iteration $k \in \{1, \cdots, N\}$, the GP prior is updated, using Bayes rule, to obtain posterior distributions conditioned on the current training set $\mathcal{S}_k = \{(\boldsymbol{x}_i, y_i)\}_{i=1}^k$ containing the past evaluated designs and observations. For any (potentially non-evaluated) design $\boldsymbol{x} \in \mathcal{X}$, the posterior mean $\overline{\mu}_k(\boldsymbol{x})$ and posterior variance $\overline{\sigma}_k^2(\boldsymbol{x})$ of the GP, conditioned on $\mathcal{S}_k$, are known in closed-form and are considered cheap to evaluate:

$$\overline{\mu}_k(\boldsymbol{x}) = K(X_k, \boldsymbol{x})^\top [K(X_k, X_k) + \lambda I]^{-1} Y_k, \tag{3}$$

$$\overline{\sigma}_k^2(\boldsymbol{x}) = \kappa(\boldsymbol{x}, \boldsymbol{x}) - K(X_k, \boldsymbol{x})^\top [K(X_k, X_k) + \lambda I]^{-1} K(X_k, \boldsymbol{x}), \tag{4}$$

where $K(X_k, X_k)$ is the $k \times k$ matrix whose $ij^{th}$ entry is $\kappa(\boldsymbol{x}_i, \boldsymbol{x}_j)$, $K(X_k, \boldsymbol{x})$ (respectively $Y_k$) is the $k \times 1$ vector whose $i^{th}$ entry is $\kappa(\boldsymbol{x}_i, \boldsymbol{x})$ (respectively $y_i$), and $\lambda$ is the noise variance. A new design $\boldsymbol{x}_{k+1}$ is then selected and evaluated with this objective function to provide an observation $y_{k+1} = f(\boldsymbol{x}_{k+1})$. This new pair $(\boldsymbol{x}_{k+1}, y_{k+1})$ is added to the current training set $\mathcal{S}_k$ to define the training set for the next iteration $\mathcal{S}_{k+1} = \mathcal{S}_k \cup \{(\boldsymbol{x}_{k+1}, y_{k+1})\}$.

In BO, the next design to evaluate is selected by solving an *auxiliary problem* (AP), typically of the form:

$$\text{(AP)} \quad \boldsymbol{x}_{k+1} = \text{argmax}_{\boldsymbol{x} \in \mathcal{X}} \, U_k(\boldsymbol{x}; \mathcal{S}_k), \tag{5}$$

where $U_k$ is a utility function to maximize. The rationale is that, because the optimization run-time or cost is dominated by the evaluation of the expensive function $f$, time and effort should be dedicated to choosing a good and informative (in a sense defined by the auxiliary problem) design to evaluate.

Solving this auxiliary problem (sometimes called maximization of an acquisition or utility function) does not involve the evaluation of the expensive objective function $f$, but only the posterior quantities of the GP and, thus, is considered cheap.

Examples of utility functions, $U_k$, used to select the next design to evaluate in Bayesian optimization include maximizing the probability of improvement (PI) [12], maximizing the expected improvement (EI) in the efficient global optimization (EGO) algorithm [10], minimizing a linear combination $\overline{\mu} - \alpha\overline{\sigma}$ of the posterior mean $\overline{\mu}$ and standard deviation $\overline{\sigma}$ in GP upper confidence bound (GP-UCB) [18], or maximizing a metric quantifying the information gain [19, 6, 7]. However, the aforementioned utility functions are oblivious to the number of objective function evaluations left and, thus, lead to greedy optimization strategies. Devising methods that account for the remaining budget would allow to better plan the sequence of designs to evaluate, balance in a principled way the exploration-exploitation trade-off encountered in BO, and thus potentially lead to performance gains.

## 3 Dynamic Programming

In this section, we review some of the key features of dynamic programming (DP) which addresses optimal decision making under uncertainty for dynamical systems. BO with a finite budget can be seen as such a problem. It has the following characteristics: (1) a statistical model to represent the objective function, (2) a system dynamic that describes how this statistical model is updated as new information is collected, and (3) a goal that can be quantified with a long-term reward. DP provides us with a mathematical formulation to address this class of problem. A full overview of DP can be found in [1, 15].

We consider a system governed by a discrete-stage dynamic. At each stage $k$, the system is fully characterized by a state $z_k \in \mathcal{Z}_k$. A control $u_k$, from a control space $\mathcal{U}_k(z_k)$, that generally depends on the state, is applied. Given a state $z_k$ and a control $u_k$, a random disturbance $w_k \in \mathcal{W}_k(z_k, u_k)$ occurs, characterized by a random variable $W_k$ with probability distribution $\mathbb{P}(\cdot|z_k, u_k)$. Then, the system evolves to a new state $z_{k+1} \in \mathcal{Z}_{k+1}$, according to the system dynamic. This can be written in the following form:

$$\forall k \in \{1, \cdots, N\}, \forall (z_k, u_k, w_k) \in \mathcal{Z}_k \times \mathcal{U}_k \times \mathcal{W}_k, \quad z_{k+1} = \mathcal{F}_k(z_k, u_k, w_k), \tag{6}$$

where $z_1$ is an initial state, $N$ is the total number of stages, or *horizon*, and $\mathcal{F}_k : \mathcal{Z}_k \times \mathcal{U}_k \times \mathcal{W}_k \mapsto \mathcal{Z}_{k+1}$ is the dynamic of the system at stage $k$ (where the spaces' dependencies are dropped for ease of notation).

We seek the construction of an optimal policy (optimal in a sense yet to define). A *policy*, $\boldsymbol{\pi} = \{\pi_1, \cdots, \pi_N\}$, is sequence of rules, $\pi_k : \mathcal{Z}_k \mapsto \mathcal{U}_k$, for $k = 1, \cdots, N$, mapping a state $z_k$ to a control $u_k = \pi_k(z_k)$.

At each stage $k$, a stage-reward function $r_k : \mathcal{Z}_k \times \mathcal{U}_k \times \mathcal{W}_k \mapsto \mathbb{R}$, quantifies the benefits of applying a control $u_k$ to a state $z_k$, subject to a disturbance $w_k$. A final reward function $r_{N+1} : \mathcal{Z}_{N+1} \mapsto \mathbb{R}$, similarly quantifies the benefits of ending at a state $z_{N+1}$. Thus, the expected reward starting from state $z_1$ and using policy $\boldsymbol{\pi}$ is:

$$J_{\boldsymbol{\pi}}(z_1) = \mathbb{E}\left[r_{N+1}(z_{N+1}) + \sum_{k=1}^{N} r_k(z_k, \pi_k(z_k), w_k)\right], \tag{7}$$

where the expectation is taken with respect to the disturbances. An optimal policy, $\boldsymbol{\pi}^*$, is a policy that maximizes this (long-term) expected reward over the set of admissible policies $\Pi$:

$$J^*(z_1) = J_{\boldsymbol{\pi}^*}(z_1) = \max_{\boldsymbol{\pi} \in \Pi} J_{\boldsymbol{\pi}}(z_1), \tag{8}$$

where $J^*$ is the optimal reward function, also called *optimal value function*. Using Bellman's *principle of optimality*, the optimal reward is given by a nested formulation and can be computed using the following DP recursive algorithm, working backward from $k = N$ to $k = 1$:

$$J_{N+1}(z_{N+1}) = r_{N+1}(z_{N+1}), \tag{9}$$
$$J_k(z_k) = \max_{u_k \in \mathcal{U}_k} \mathbb{E}[r_k(z_k, u_k, w_k) + J_{k+1}(\mathcal{F}_k(z_k, u_k, w_k))]. \tag{10}$$

The optimal reward $J^*(z_1)$ is then given by $J_1(z_1)$, and if $u_k^* = \pi_k^*(z_k)$ maximizes the right hand side of Eq. 10 for all $k$ and all $z_k$, then the policy $\boldsymbol{\pi}^* = \{\pi_1^*, \cdots, \pi_N^*\}$ is optimal (e.g., [1], p.23).

# 4 Bayesian Optimization with a Finite Budget

In this section, we define the auxiliary problem of BO with a finite budget (Eq. 5) as a DP instance.

At each iteration $k$, we seek to evaluate the design that will lead, once the evaluation budget $N$ has been consumed, to the maximum reduction of the objective function. In general, the value of the objective function $f(\boldsymbol{x})$ at a design $\boldsymbol{x}$ is unknown before its evaluation and, thus, estimating the long-term effect of an evaluation is not possible. However, using the GP representing $f$, it is possible to characterize the unknown $f(\boldsymbol{x})$ by a distribution. This can be used to simulate sequences of designs and function values (i.e., optimization scenarios), compute their rewards and associated probabilities, without evaluating $f$. Using this simulation machinery, it is possible to capture the goal of achieving a long term reward in a utility function $U_k$. We now formulate the simulation of optimization scenarios in the DP context and proceed with the definition of such utility function $U_k$.

We consider that the process of optimization is a dynamical system. At each iteration $k$, this system is fully characterized by a state $z_k$ equal to the training set $\mathcal{S}_k$. The system is actioned by a control $u_k$ equal to the design $\boldsymbol{x}_{k+1}$ selected to be evaluated. For a given state and control, the value of the objective function is unknown and modeled as a random variable $W_k$, characterized by:

$$W_k \sim \mathcal{N}\left(\overline{\mu}_k(\boldsymbol{x}_{k+1}), \overline{\sigma}_k^2(\boldsymbol{x}_{k+1})\right), \tag{11}$$

where $\overline{\mu}_k(\boldsymbol{x}_{k+1})$ and $\overline{\sigma}_k^2(\boldsymbol{x}_{k+1})$ are the posterior mean and variance of the GP at $\boldsymbol{x}_{k+1}$, conditioned on $\mathcal{S}_k$. We define a disturbance $w_k$ to be equal to a realization $f_{k+1}$ of $W_k$. Thus, $w_k = f_{k+1}$ represents a possible (simulated) value of the objective function at $\boldsymbol{x}_{k+1}$. Note that this simulated value of the objective function, $f_{k+1}$, is not the value of the objective function $y_{k+1} = f(\boldsymbol{x}_{k+1})$. Hence, we have the following identities: $\mathcal{Z}_k = (\mathcal{X} \times \mathbb{R})^k, \mathcal{U}_k = \mathcal{X}$ and $\mathcal{W}_k = \mathbb{R}$.

The new state $z_{k+1}$ is then defined to be the augmented training set $\mathcal{S}_{k+1} = \mathcal{S}_k \cup \{(\boldsymbol{x}_{k+1}, f_{k+1})\}$, and the system dynamic can be written as:

$$\mathcal{S}_{k+1} = \mathcal{F}_k(\mathcal{S}_k, \boldsymbol{x}_{k+1}, f_{k+1}) = \mathcal{S}_k \cup \{(\boldsymbol{x}_{k+1}, f_{k+1})\}. \tag{12}$$

The disturbances $w_{k+1}$ at iteration $k+1$ are then characterized, using Bayes' rule, by the posterior of the GP conditioned on the training set $\mathcal{S}_{k+1}$.

To optimally control this system (i.e., to use an optimal strategy to solve OP), we define the stage-reward function at iteration $k$ to be the reduction in the objective function obtained at stage $k$:

$$r_k(\mathcal{S}_k, \boldsymbol{x}_{k+1}, f_{k+1}) = \max\left\{0, f_{min}^{\mathcal{S}_k} - f_{k+1}\right\}, \tag{13}$$

where $f_{min}^{\mathcal{S}_k}$ is the minimum value of the objective function in the training set $\mathcal{S}_k$. We define the final reward to be zero: $r_{N+1}(\mathcal{S}_{N+1}) = 0$. The utility function, at a given iteration $k$ characterized by $\mathcal{S}_k$, is defined to be the expected reward:

$$\forall \boldsymbol{x}_{k+1} \in \mathcal{X}, \ U_k(\boldsymbol{x}_{k+1}; \mathcal{S}_k) = \mathbb{E}[r_k(\mathcal{S}_k, \boldsymbol{x}_{k+1}, f_{k+1}) + J_{k+1}(\mathcal{F}_k(\mathcal{S}_k, \boldsymbol{x}_{k+1}, f_{k+1}))], \tag{14}$$

where the expectation is taken with respect to the disturbances, and $J_{k+1}$ is defined by Eqs. 9-10. Note that $\mathbb{E}[r_k(\mathcal{S}_k, \boldsymbol{x}_{k+1}, f_{k+1})]$ is simply the expected improvement given, for all $\boldsymbol{x} \in \mathcal{X}$, by:

$$EI(\boldsymbol{x}; \mathcal{S}_k) = \left(f_{min}^{\mathcal{S}_k} - \overline{\mu}_k(\boldsymbol{x})\right) \Phi\left(\frac{f_{min}^{\mathcal{S}_k} - \overline{\mu}_k(\boldsymbol{x})}{\overline{\sigma}_k(\boldsymbol{x})}\right) + \overline{\sigma}_k(\boldsymbol{x})\phi\left(\frac{f_{min}^{\mathcal{S}_k} - \overline{\mu}_k(\boldsymbol{x})}{\overline{\sigma}_k(\boldsymbol{x})}\right), \tag{15}$$

where $\Phi$ is the standard Gaussian CDF and $\phi$ is the standard Gaussian PDF.

In other words, the GP is used to simulate possible scenarios, and the next design to evaluate is chosen to maximize the decrease of the objective function, over the remaining iterations, averaged over all possible simulated scenarios.

Several related methods have been proposed to go beyond greedy BO strategies. Optimal formulations for BO with a finite budget have been explored in [14, 4]. Both formulations involve nested maximizations and expectations. Those authors note that their $N$-steps lookahead methods scale poorly with the number of steps considered (i.e., the budget $N$); they are able to solve the problem for two-steps lookahead. For some specific instances of BO (e.g., finding the super-level set of a one-dimensional function), the optimal multi-step strategy can be computed efficiently [3]. Approximation techniques accounting for more steps have been recently proposed. They leverage partial

tree exploration [13] or Lipschitz reward function [11] and have been applied to cases where the control spaces $\mathcal{U}_k$ are finite (e.g., at each iteration, $u_k$ is one of the 4 or 8 directions that a robot can take to move before it evaluates $f$). Theoretical performance guarantees are provided for the algorithm proposed in [11]. Another approximation technique for non-greedy BO has been proposed in GLASSES [5] and is applicable to uncountable control space $\mathcal{U}_k$. It builds an approximation of the $N$-steps lookahead formulation by using a one-step lookahead algorithm with approximation of the value function $J_{k+1}$. The approximate value function is induced by a heuristic oracle based on a batch Bayesian optimization method. The oracle is used to select up to 15 steps at once to approximate the value function.

In this paper, we propose to use rollout, an ADP algorithm, to address the intractability of the DP formulation. The proposed approach is not restricted to countable control spaces, and accounts for more than two steps. This is achieved by approximating the value function $J_{k+1}$ with simulations over several steps, where the information acquired at each simulated step is explicitly used to simulate the next step. Note that this is a closed-loop approach, in comparison to GLASSES [5] which is an open-loop approach. In contrast to the DP formulation, the decision made at each simulated step of the rollout is not optimal, but guided by problem-dependent heuristics. In this paper we propose the use of heuristics adapted to BO, leveraging existing greedy BO strategies.

## 5    Rollout for Bayesian Optimization

Solving the auxiliary problem defined by Eqs. 5,14 is challenging. It requires the solution of nested maximizations and expectations for which there is no closed-form expression known. In finite spaces, the DP algorithm already suffers from the curse of dimensionality. In this particular setting, the state spaces $\mathcal{Z}_k = (\mathcal{X} \times \mathbb{R})^k$ are uncountable and their dimension increases by $d + 1$ at each stage. The control spaces $\mathcal{U}_k = \mathcal{X}$ are also uncountable, but of fixed dimension. Thus, solving Eq. 5 with utility function defined by Eq. 14 is intractable.

To simplify the problem, we use ADP to approximate $U_k$ with the rollout algorithm (see [1, 15] for an overview). It is a one-step lookahead technique where $J_{k+1}$ is approximated using simulations over several future steps. The difference with the DP formulation is that, in those simulated future steps, rollout relaxes the requirement to optimally select a design (which is the origin of the nested maximizations). Instead, rollout uses a suboptimal heuristic to decide which control to apply for a given state. This suboptimal heuristic is problem-dependent and, in the context of BO with a finite budget, we propose to use existing greedy BO algorithms as such a heuristic. Our algorithm proceeds as follows.

For any iteration $k$, the optimal reward to go, $J_{k+1}$ (Eq. 14), is approximated by $H_{k+1}$, the reward to go induced by a heuristic $\boldsymbol{\pi} = (\pi_1, \cdots, \pi_N)$, also called *base policy*. $H_{k+1}$ is recursively given by:

$$H_N(\mathcal{S}_N) = EI(\pi_N(\mathcal{S}_N); \mathcal{S}_N), \tag{16}$$

$$H_n(\mathcal{S}_n) = \mathbb{E}\left[r_n(\mathcal{S}_n, \pi_n(\mathcal{S}_n), f_{n+1}) + \gamma H_{n+1}(\mathcal{F}(\mathcal{S}_n, \pi_n(\mathcal{S}_n), f_{n+1}))\right], \tag{17}$$

for all $n \in \{k+1, \cdots, N-1\}$, where $\gamma \in [0, 1]$ is a discount factor incentivizing the early collection of reward. A discount factor $\gamma = 0$, leads to a greedy strategy that maximizes the immediate collection of reward. This corresponds to maximizing the EI. On the other hand, $\gamma = 1$, means that there is no differentiation between collecting reward early or late in the optimization. Note that $H_{k+1}$ is defined by recursion, and involves nested expectations. However, the nested maximizations are replaced by the use of the base policy $\boldsymbol{\pi}$. An important point is that, even if its definition is recursive, $H_{k+1}$ can be computed in a forward manner, unlike $J_{k+1}$ which has to be computed in a backward fashion (see Eqs. 9,10). The DP and the rollout formulations are illustrated in Fig.1. The approximated reward $H_{k+1}$ is then numerically approximated by $\widetilde{H}_{k+1}$ using several simplifications. First, we use a rolling horizon, $h$, to alleviate the curse of dimensionality. At a given iteration $k$, a rolling horizon limits the number of stages considered to compute the approximate reward to go by replacing the horizon $N$ by $\tilde{N} = \min\{k + h, N\}$. Second, expectations are taken with respect to the (Gaussian) disturbances and are approximated using Gauss-Hermite quadrature. We obtain the following formulation:

$$\tilde{H}_{\tilde{N}}(\mathcal{S}_{\tilde{N}}) = EI(\pi_{\tilde{N}}(\mathcal{S}_{\tilde{N}}); \mathcal{S}_{\tilde{N}}), \tag{18}$$

$$\widetilde{H}_n(\mathcal{S}_n) = \sum_{q=1}^{N_q} \alpha^{(q)} \left[ r_n\left(\mathcal{S}_n, \pi_n(\mathcal{S}_n), f_{n+1}^{(q)}\right) + \gamma \widetilde{H}_{n+1}\left(\mathcal{F}\left(\mathcal{S}_n, \pi_n(\mathcal{S}_n), f_{n+1}^{(q)}\right)\right) \right], \tag{19}$$

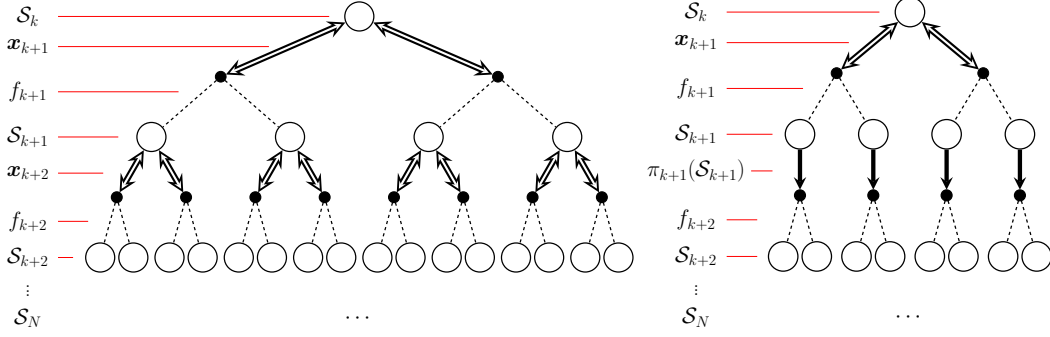

Figure 1: Graphs representing the DP (left) and the rollout (right) formulations (in the binary decisions, binary disturbances case). Each white circle represents a training set, each black circle represents a training set and a design. Double arrows are decisions that depend on decisions lower in the graph (leading to nested optimizations in the DP formulation), single arrows represent decisions made using a heuristic (independent of the lower part of the graph). Dashed lines are simulated values of the objective function and lead to the computation of expectations. Note the simpler structure of the rollout graph compared to the DP one.

for all $n \in \{k+1, \cdots, \tilde{N}-1\}$, where $N_q \in \mathbb{N}$ is the number of quadrature weights $\alpha^{(q)} \in \mathbb{R}$ and points $f_{k+1}^{(q)} \in \mathbb{R}$, and $r_k$ is the stage-reward defined by Eq. 13. Finally, for all iterations $k \in \{1, \cdots, N-1\}$ and for all $\boldsymbol{x}_{k+1} \in \mathcal{X}$, we define the utility function to be:

$$U_k(\boldsymbol{x}_{k+1}; \mathcal{S}_k) = \sum_{q=1}^{N_q} \alpha^{(q)} \left[ r_k \left( \mathcal{S}_k, \boldsymbol{x}_{k+1}, f_{k+1}^{(q)} \right) + \gamma \widetilde{H}_{k+1} \left( \mathcal{F} \left( \mathcal{S}_k, \boldsymbol{x}_{k+1}, f_{k+1}^{(q)} \right) \right) \right]. \quad (20)$$

We note that for the last iteration, $k = N$, the utility function is known in closed form:

$$U_N(\boldsymbol{x}_{N+1}; \mathcal{S}_N) = EI(\boldsymbol{x}_{N+1}; \mathcal{S}_N). \quad (21)$$

The base policy $\boldsymbol{\pi}$ used as a heuristic in the rollout is problem-dependent. A good heuristic $\boldsymbol{\pi}$ should be cheap to compute and induce an expected reward $J_{\boldsymbol{\pi}}$ close to the optimal expected reward $J_{\boldsymbol{\pi}^*}$ (Eq. 7). In the context of BO with a finite budget, this heuristic should mimic an optimal strategy that balances the exploration-exploitation trade-off. We propose to use existing BO strategies, in particular, maximization of the expected improvement (which has an exploratory behavior) and minimization of the posterior mean (which has an exploitation behavior) to build the base policy. For every iteration $k \in \{1, \cdots, N-1\}$, we define $\boldsymbol{\pi} = \{\pi_{k+1}, \cdots, \pi_{\tilde{N}}\}$ such that, at stage $n \in \{k+1, \tilde{N}-1\}$, the policy component, $\pi_n : \mathcal{Z}_n \mapsto \mathcal{X}$, maps a state $z_n = \mathcal{S}_n$ to the design $\boldsymbol{x}_{n+1}$ that maximizes the expected improvement (Eq. 15):

$$\boldsymbol{x}_{n+1} = \underset{\boldsymbol{x} \in \mathcal{X}}{\operatorname{argmax}} \, EI(\boldsymbol{x}; \mathcal{S}_n). \quad (22)$$

The last policy component, $\pi_{\tilde{N}} : \mathcal{Z}_{\tilde{N}} \mapsto \mathcal{X}$, is defined to map a state $z_{\tilde{N}} = \mathcal{S}_{\tilde{N}}$ to the design $\boldsymbol{x}_{\tilde{N}+1}$ that minimizes the posterior mean (Eq. 3):

$$\boldsymbol{x}_{\tilde{N}+1} = \underset{\boldsymbol{x} \in \mathcal{X}}{\operatorname{argmin}} \, \overline{\mu}_{\tilde{N}}(\boldsymbol{x}). \quad (23)$$

Each evaluation of the utility function requires $\mathcal{O}\left(N_q^h\right)$ applications of a heuristic. In our approach, the heuristic involves optimizing a quantity that requires $\mathcal{O}\left(|\mathcal{S}_k|^2\right)$ of work (rank-1 update of the Cholesky decomposition to update the GP, and back-substitution for the posterior variance).

To summarize, we propose to use rollout, a one-step lookahead algorithm that approximates $J_{k+1}$. This approximation is computed using simulation over several steps (e.g., more than 3 steps), where the information collected after a simulated step is explicitly used to simulate the next step (i.e., it is a closed-loop approach). This is achieved using a heuristic instead of the optimal strategy, and thus, leads to a formulation without nested maximizations.

# 6   Experiments and Discussion

In this section, we apply the proposed algorithm to several optimization problems with a finite budget and demonstrate its performance on GP generated and classic test functions.

We use a zero-mean GP with square-exponential kernel (hyper-parameters: maximum variance $\sigma^2 = 4$, length scale $L = 0.1$, noise variance $\lambda = 10^{-3}$) to generate 24 objective functions defined on $\mathcal{X} = [0,1]^2$. We generate 10 designs from a uniform distribution on $\mathcal{X}$, and use them as 10 different initial guesses for optimization. Thus, for each optimization, the initial training set $\mathcal{S}_1$ contains one training point. All algorithms are given a budget of $N = 15$ evaluations. For each of the initial guess and each objective function, we run the BO algorithm with the following utility functions: PI, EI and GP-UCB (with the parameter balancing exploration and exploitation set to $\alpha = 3$). We also run the rollout algorithm proposed in Sec. 5 and defined by Eqs. 5,20, for the same objective functions and with the same initial guesses for different parameters of the rolling horizon $h \in \{2,3,4,5\}$ and discount factor $\gamma \in \{0.5, 0.7, 0.9, 1.0\}$. All algorithms use the same kernel and hyper-parameters as those used to generate the objective functions.

Given a limited evaluation budget, we evaluate the performance of an algorithm for the original problem (Eq. 1) in terms of gap $G$ [8]. The gap measures the best decrease in objective function from the first to the last iteration, normalized by the maximum reduction possible:

$$G = \frac{f_{min}^{\mathcal{S}_1} - f_{min}^{\mathcal{S}_{N+1}}}{f_{min}^{\mathcal{S}_1} - f(\boldsymbol{x}^*)}. \tag{24}$$

The mean and the median performances of the rollout algorithm are computed for the 240 experiments for the 16 configurations of discount factors and rolling horizons. The results are reported in Table 1.

Table 1: Mean (left) and median (right) performance $G$ over 24 objective functions and 10 initial guesses for different rolling horizons $h$ and discount factors $\gamma$.

| $\gamma$ | $h=2$ | $h=3$ | $h=4$ | $h=5$ | $\gamma$ | $h=2$ | $h=3$ | $h=4$ | $h=5$ |
|---|---|---|---|---|---|---|---|---|---|
| 0.5 | 0.790 | 0.811 | 0.799 | 0.817 | 0.5 | 0.849 | 0.862 | 0.858 | 0.856 |
| 0.7 | 0.787 | 0.786 | 0.787 | 0.836 | 0.7 | 0.849 | 0.830 | 0.806 | 0.878 |
| 0.9 | 0.816 | 0.767 | 0.827 | 0.828 | 0.9 | 0.896 | 0.839 | 0.876 | 0.850 |
| 1.0 | 0.818 | 0.793 | 0.842 | 0.812 | 1.0 | 0.870 | 0.861 | 0.917 | 0.858 |

The mean gap achieved is $G = 0.698$ for PI, $G = 0.762$ for EI and $G = 0.711$ for GP-UCB. All the configurations of the rollout algorithm outperform the three greedy BO algorithms. The best performance is achieved by the configuration $\gamma = 1.0$ and $h = 4$. For this configuration, the performance increase with respect to EI is about $8\%$. The worst mean configuration ($\gamma = 0.9$ and $h = 3$) still outperforms EI by $0.5\%$.

The median performance achieved is $G = 0.738$ for PI, $G = 0.777$ for EI and $G = 0.770$ for GP-UCB. All the configurations of the rollout algorithm outperform the three greedy BO algorithms. The best performance is achieved by the configuration $\gamma = 1.0$ and $h = 4$ (same as best mean performance). For this configuration, the performance increase with respect to EI is about $14\%$. The worst rollout configuration ($\gamma = 0.7$ and $h = 4$) still outperforms EI by $2.9\%$. The complete distribution of gaps achieved by the greedy BO algorithms and the best and worst configurations of the rollout is shown in Fig. 2.

We notice that increasing the length of the rolling horizon does not necessarily increase the gap (see Table 1). This is a classic result from DP (Sec. 6.5.1 of [1]). We also notice that discounting the future rewards has no clear effect on the gap. For all discount factors tested, we notice that reward is not only collected at the last stage (See Fig. 2). This is a desirable property. Indeed, in a case where the optimization has to be stopped before the end of the budget is reached, one would wish to have collected part of the reward.

We now evaluate the performance on test functions.[1] We consider four rollout configurations R-4-9 ($h = 4$, $\gamma = 0.9$), R-4-10 ($h = 4$, $\gamma = 1.0$), R-5-9 ($h = 5$, $\gamma = 0.9$) and R-5-10 ($h = 5$, $\gamma = 1.0$)

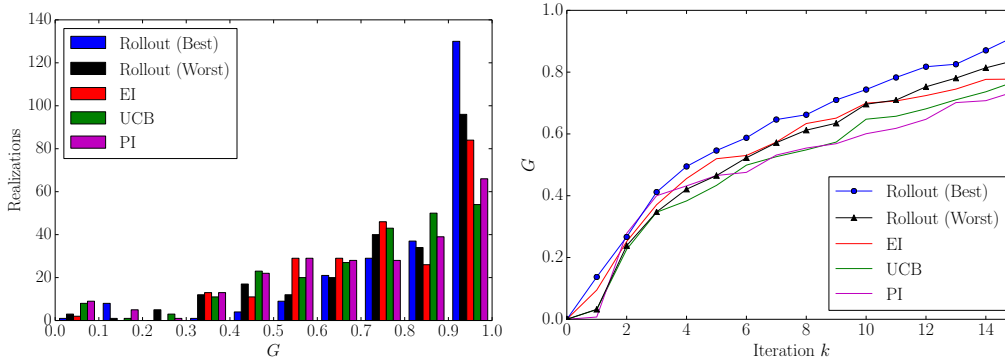

Figure 2: Left: Histogram of gap for the rollout (best and worst mean configurations tested) and greedy BO algorithms. Right: Median gap of the rollout (for the best and worst mean configurations tested) and other algorithms as a function of iteration (budget of $N = 15$).

and two additional BO algorithms: PES [7] and the non-greedy GLASSES [5]. We use a square-exponential kernel for each algorithm (hyper-parameters: maximum variance $\sigma^2 = 4$, noise variance $\lambda = 10^{-3}$, length scale $L$ set to $10\%$ of the design space length scale). We generate 40 designs from a uniform distribution on $\mathcal{X}$, and use them as 40 different initial guesses for optimization. Each algorithm is given $N = 15$ evaluations. The mean and median gap (over the 40 initial guesses) for each function define 8 metrics (shown in Table 2). We found that rollout had the best metric 3 times out of 8, and was never the worst algorithm. PES was found to perform best on 3 metrics out of 8 but was the worst algorithm for 2 metrics out of 8. GLASSES was never the best algorithm and performed the worst in one metric. Note that the rollout configuration R-4-9 outperforms GLASSES on 5 metrics out of 6 (excluding the case of the Griewank function). Thus, our rollout algorithm performs well and shows robustness.

Table 2: Mean and median gap $G$ over 40 initial guesses.

| Function name | | PI | EI | UCB | PES | GLASSES | R-4-9 | R-4-10 | R-5-9 | R-5-10 |
|---|---|---|---|---|---|---|---|---|---|---|
| Branin-Hoo | Mean | 0.847 | *0.818* | 0.848 | 0.861 | 0.846 | **0.904** | 0.898 | 0.887 | 0.903 |
| | Median | 0.922 | *0.909* | 0.910 | **0.983** | *0.909* | 0.959 | 0.943 | 0.921 | 0.950 |
| Goldstein-Price | Mean | 0.873 | 0.866 | *0.733* | 0.819 | 0.782 | **0.895** | 0.784 | 0.861 | 0.743 |
| | Median | 0.983 | 0.981 | *0.899* | 0.987 | 0.919 | **0.991** | 0.985 | 0.989 | 0.928 |
| Griewank | Mean | *0.827* | 0.884 | 0.913 | **0.972** | 1$^2$ | 0.882 | 0.885 | 0.930 | 0.867 |
| | Median | *0.904* | 0.953 | 0.970 | **0.987** | 1$^2$ | 0.967 | 0.962 | 0.960 | 0.954 |
| Six-hump Camel | Mean | 0.850 | **0.887** | 0.817 | *0.664* | 0.776 | 0.860 | 0.825 | 0.793 | 0.803 |
| | Median | 0.893 | **0.970** | 0.915 | *0.801* | 0.941 | 0.926 | 0.900 | 0.941 | 0.907 |

# 7 Conclusions

We presented a novel algorithm to perform Bayesian optimization with a finite budget of evaluations. The next design to evaluate is chosen to maximize a utility function that quantifies long-term rewards. We propose to employ an approximate dynamic programming algorithm, rollout, to approximate this utility function. Rollout leverages heuristics to circumvent the need for nested maximizations. We propose to build such a heuristic using existing suboptimal Bayesian optimization strategies, in particular maximization of the expected improvement and minimization of the posterior mean. The proposed approximate dynamic programming algorithm is empirically shown to outperform popular greedy and non-greedy Bayesian optimization algorithms on multiple test cases.

This work was supported in part by the AFOSR MURI on multi-information sources of multi-physics systems under Award Number FA9550-15-1-0038, program manager Dr. Jean-Luc Cambier.

---

$^2$This gap $G = 1$ results from an arbitrary choice made by one optimizer used by GLASSES to evaluate the origin. The origin happens to be the minimizer of the Griewank function. We thus exclude those results from the analysis.

## Footnotes

[1]Test functions from `http://www.sfu.ca/~ssurjano/optimization.html`.

# References

[1] D. P. Bertsekas. *Dynamic programming and optimal control*, volume 1. Athena Scientific, 1995.

[2] E. Brochu, V. M. Cora, and N. De Freitas. A tutorial on Bayesian optimization of expensive cost functions, with application to active user modeling and hierarchical reinforcement learning. *arXiv preprint arXiv:1012.2599*, 2010.

[3] J. M. Cashore, L. Kumarga, and P. I. Frazier. Multi-step Bayesian optimization for one-dimensional feasibility determination. Working paper. Retrieved from https://people.orie.cornell.edu/pfrazier/pub/workingpaper-CashoreKumargaFrazier.pdf.

[4] D. Ginsbourger and R. Le Riche. Towards Gaussian process-based optimization with finite time horizon. In *mODa 9–Advances in Model-Oriented Design and Analysis*, pages 89–96. Springer, 2010.

[5] J. González, M. Osborne, and N. D. Lawrence. GLASSES: Relieving the myopia of Bayesian optimisation. In *Proceedings of the 19th International Conference on Artificial Intelligence and Statistics*, pages 790–799, 2016.

[6] P. Hennig and C. J. Schuler. Entropy search for information-efficient global optimization. *The Journal of Machine Learning Research*, 13(1):1809–1837, 2012.

[7] J. M. Hernández-Lobato, M. W. Hoffman, and Z. Ghahramani. Predictive entropy search for efficient global optimization of black-box functions. In *Advances in Neural Information Processing Systems*, pages 918–926, 2014.

[8] D. Huang, T. T. Allen, W. I. Notz, and N. Zeng. Global optimization of stochastic black-box systems via sequential kriging meta-models. *Journal of Global Optimization*, 34(3):441–466, 2006.

[9] D. R. Jones. A taxonomy of global optimization methods based on response surfaces. *Journal of Global Optimization*, 21(4):345–383, 2001.

[10] D. R. Jones, M. Schonlau, and W. J. Welch. Efficient global optimization of expensive black-box functions. *Journal of Global Optimization*, 13(4):455–492, 1998.

[11] C. K. Ling, K. H. Low, and P. Jaillet. Gaussian process planning with lipschitz continuous reward functions: Towards unifying bayesian optimization, active learning, and beyond. In *30th AAAI Conference on Artificial Intelligence*, 2016.

[12] D. J. Lizotte. *Practical Bayesian Optimization*. PhD thesis, Edmonton, Alta., Canada, 2008. AAINR46365.

[13] R. Marchant, F. Ramos, and S. Sanner. Sequential Bayesian optimisation for spatial-temporal monitoring. 2015.

[14] M. A. Osborne, R. Garnett, and S. J. Roberts. Gaussian processes for global optimization. In *3rd International Conference on Learning and Intelligent Optimization (LION3)*, pages 1–15, 2009.

[15] W. B. Powell. *Approximate Dynamic Programming: Solving the Curses of Dimensionality*, volume 842. John Wiley & Sons, 2011.

[16] C. E. Rasmussen and C. K. I. Williams. *Gaussian Processes for Machine Learning*. MIT Press, Cambridge, MA, 2006.

[17] J. Snoek, H. Larochelle, and R. P. Adams. Practical Bayesian optimization of machine learning algorithms. In *Advances in Neural Information Processing Systems*, pages 2951–2959, 2012.

[18] N. Srinivas, A. Krause, S. M. Kakade, and M. Seeger. Gaussian process optimization in the bandit setting: No regret and experimental design. In *Proceedings of the 27th International Conference on Machine Learning*, pages 1015–1022, 2010.

[19] J. Villemonteix, E. Vazquez, and E. Walter. An informational approach to the global optimization of expensive-to-evaluate functions. *Journal of Global Optimization*, 44(4):509–534, 2009.

